# Learning a Discriminative Hidden Part Model for Human Action Recognition

**Yang Wang**
School of Computing Science
Simon Fraser University
Burnaby, BC, Canada, V5A 1S6
ywang12@cs.sfu.ca

**Greg Mori**
School of Computing Science
Simon Fraser University
Burnaby, BC, Canada, V5A 1S6
mori@cs.sfu.ca

## Abstract

We present a discriminative part-based approach for human action recognition from video sequences using motion features. Our model is based on the recently proposed hidden conditional random field (hCRF) for object recognition. Similar to hCRF for object recognition, we model a human action by a flexible constellation of parts conditioned on image observations. Different from object recognition, our model combines both large-scale global features and local patch features to distinguish various actions. Our experimental results show that our model is comparable to other state-of-the-art approaches in action recognition. In particular, our experimental results demonstrate that combining large-scale global features and local patch features performs significantly better than directly applying hCRF on local patches alone.

## 1  Introduction

Recognizing human actions from videos is a task of obvious scientific and practical importance. In this paper, we consider the problem of recognizing human actions from video sequences on a frame-by-frame basis. We develop a discriminatively trained hidden part model to represent human actions. Our model is inspired by the hidden conditional random field (hCRF) model [16] in object recognition.

In object recognition, there are three major representations: global template (rigid, e.g. [3], or deformable, e.g. [1]), bag-of-words [18], and part-based [7, 6]. All three representations have been shown to be effective on certain object recognition tasks. In particular, recent work [6] has shown that part-based models outperform global templates and bag-of-words on challenging object recognition tasks.

A lot of the ideas used in object recognition can also be found in action recognition. For example, there is work [2] that treats actions as space-time shapes and reduces the problem of action recognition to 3D object recognition. In action recognition, both global template [5] and bag-of-words models [14, 4, 15] have been shown to be effective on certain tasks. Although conceptually appealing and promising, the merit of part-based models has not yet been widely recognized in action recognition. The goal of this work is to address this gap.

Our work is partly inspired by a recent work in part-based event detection [10]. In that work, template matching is combined with a pictorial structure model to detect and localize actions in crowded videos. One limitation of that work is that one has to manually specify the parts. Unlike Ke et al. [10], the parts in our model are initialized automatically.

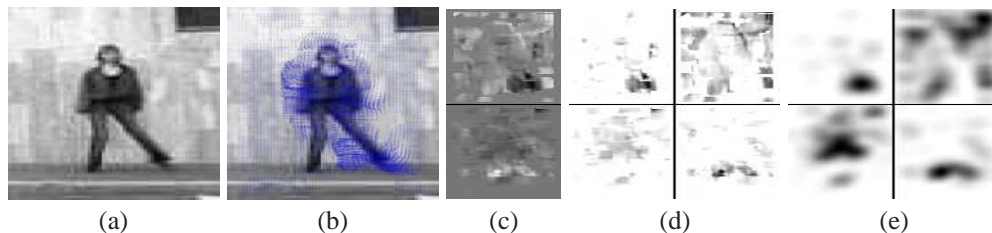

| (a) | (b) | (c) | (d) | (e) |

Figure 1: Construction of the motion descriptor. (a) original image; (b) optical flow; (c) $x$ and $y$ components of optical flow vectors $F_x$, $F_y$; (d) half-wave rectification of $x$ and $y$ components to obtain 4 separate channels $F_x^+, F_x^-, F_y^+, F_y^-$; (e) final blurry motion descriptors $Fb_x^+, Fb_x^-, Fb_y^+, Fb_y^-$.

The major contribution of this work is that we combine the flexibility of part-based approaches with the global perspectives of large-scale template features in a discriminative model. We show that the combination of part-based and large-scale template features improves the final results.

## 2 Our Model

The hidden conditional random field model [16] was originally proposed for object recognition and has also been applied in sequence labeling [19]. Objects are modeled as flexible constellations of parts conditioned on the appearances of local patches found by interest point operators. The probability of the assignment of parts to local features is modeled by a conditional random field (CRF) [11]. The advantage of the hCRF is that it relaxes the conditional independence assumption commonly used in the bag-of-words approaches of object recognition.

Similarly, local patches can also be used to distinguish actions. Figure. 4(a) shows some examples of human motion and the local patches that can be used to distinguish them. A bag-of-words representation can be used to model these local patches for action recognition. However, it suffers from the same restriction of conditional independence assumption that ignores the spatial structures of the parts. In this work, we use a variant of hCRF to model the constellation of these local patches in order to alleviate this restriction.

There are also some important differences between objects and actions. For objects, local patches could carry enough information for recognition. But for actions, we believe local patches are not sufficiently informative. In our approach, we modify the hCRF model to combine local patches and large-scale global features. The large-scale global features are represented by a root model that takes the frame as a whole. Another important difference with [16] is that we use the learned root model to find discriminative local patches, rather than using a generic interest-point operator.

### 2.1 Motion features

Our model is built upon the optical flow features in [5]. This motion descriptor has been shown to perform reliably with noisy image sequences, and has been applied in various tasks, such as action classification, motion synthesis, etc.

To calculate the motion descriptor, we first need to track and stabilize the persons in a video sequence. Any reasonable tracking or human detection algorithm can be used, since the motion descriptor we use is very robust to jitters introduced by the tracking. Given a stabilized video sequence in which the person of interest appears in the center of the field of view, we compute the optical flow at each frame using the Lucas-Kanade [12] algorithm. The optical flow vector field $F$ is then split into two scalar fields $F_x$ and $F_y$, corresponding to the $x$ and $y$ components of $F$. $F_x$ and $F_y$ are further half-wave rectified into four non-negative channels $F_x^+, F_x^-, F_y^+, F_y^-$, so that $F_x = F_x^+ - F_x^-$ and $F_y = F_y^+ - F_y^-$. These four non-negative channels are then blurred with a Gaussian kernel and normalized to obtain the final four channels $Fb_x^+, Fb_x^-, Fb_y^+, Fb_y^-$ (see Fig. 1).

## 2.2 Hidden conditional random field(hCRF)

Now we describe how we model a frame $I$ in a video sequence. Let $\mathbf{x}$ be the motion feature of this frame, and $y$ be the corresponding class label of this frame, ranging over a finite label alphabet $\mathcal{Y}$. Our task is to learn a mapping from $\mathbf{x}$ to $y$. We assume each image $I$ contains a set of salient patches $\{I_1, I_2, ..., I_m\}$. we will describe how to find these salient patches in Sec. 3. Our training set consists of labeled images $(\mathbf{x}^t, y^t)$ (as a notation convention, we use superscripts to index training images and subscripts to index patches) for $t = 1, 2, ..., n$, where $y^t \in \mathcal{Y}$ and $\mathbf{x}^t = (x_1^t, x_2^t..., x_m^t)$. $x_i^t = \mathbf{x}^t(I_i^t)$ is the feature vector extracted from the global motion feature $\mathbf{x}^t$ at the location of the patch $I_i^t$. For each image $I = \{I_1, I_2, ..., I_m\}$, we assume there exists a vector of hidden "part" variables $\mathbf{h} = \{h_1, h_2, ..., h_m\}$, where each $h_i$ takes values from a finite set $\mathcal{H}$ of possible parts. Intuitively, each $h_i$ assigns a part label to the patch $I_i$, where $i = 1, 2, ..., m$. For example, for the action "waving-two-hands", these parts may be used to characterize the movement patterns of the left and right arms. The values of $\mathbf{h}$ are not observed in the training set, and will become the hidden variables of the model.

We assume there are certain constraints between some pairs of $(h_j, h_k)$. For example, in the case of "waving-two-hands", two patches $h_j$ and $h_k$ at the left hand might have the constraint that they tend to have the same part label, since both of them are characterized by the movement of the left hand. If we consider $h_i(i = 1, 2, ..., m)$ to be vertices in a graph $G = (E, V)$, the constraint between $h_j$ and $h_k$ is denoted by an edge $(j, k) \in E$. See Fig. 2 for an illustration of our model. Note that the graph structure can be different for different images. We will describe how to find the graph structure $E$ in Sec. 3.

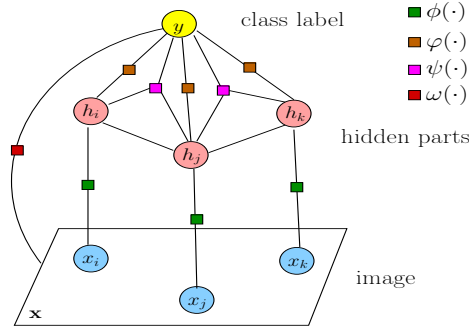

Figure 2: Illustration of the model. Each circle corresponds to a variable, and each square corresponds to a factor in the model.

Given the motion feature $\mathbf{x}$ of an image $I$, its corresponding class label $y$, and part labels $\mathbf{h}$, a hidden conditional random field is defined as $p(y, \mathbf{h}|\mathbf{x}; \theta) = \frac{\exp(\Psi(y, \mathbf{x}, \mathbf{h}; \theta))}{\sum_{\hat{y} \in \mathcal{Y}} \sum_{\hat{\mathbf{h}} \in \mathcal{H}^m} \exp(\Psi(\hat{y}, \mathbf{x}, \hat{\mathbf{h}}; \theta))}$, where $\theta$ is the model parameter, and $\Psi(y, \mathbf{h}, \mathbf{x}; \theta) \in \mathbb{R}$ is a potential function parameterized by $\theta$. It follows that

$$p(y|\mathbf{x}; \theta) = \sum_{\mathbf{h} \in \mathcal{H}^m} p(y, \mathbf{h}|\mathbf{x}; \theta) = \frac{\sum_{\mathbf{h} \in \mathcal{H}^m} \exp(\Psi(y, \mathbf{h}, \mathbf{x}; \theta))}{\sum_{\hat{y} \in \mathcal{Y}} \sum_{\mathbf{h} \in \mathcal{H}^m} \exp(\Psi(\hat{y}, \mathbf{h}, \mathbf{x}; \theta))} \tag{1}$$

We assume $\Psi(y, \mathbf{h}, \mathbf{x})$ is linear in the parameters $\theta = \{\alpha, \beta, \gamma, \eta\}$:

$$\Psi(y, \mathbf{h}, \mathbf{x}; \theta) = \sum_{j \in V} \alpha^\top \cdot \phi(x_j, h_j) + \sum_{j \in V} \beta^\top \cdot \varphi(y, h_j) + \sum_{(j,k) \in E} \gamma^\top \cdot \psi(y, h_j, h_k) + \eta^\top \cdot \omega(y, \mathbf{x}) \tag{2}$$

where $\phi(\cdot)$ and $\varphi(\cdot)$ are feature vectors depending on unary $h_j$'s, $\psi(\cdot)$ is a feature vector depending on pairs of $(h_j, h_k)$, $\omega(\cdot)$ is a feature vector that does not depend on the values of hidden variables. The details of these feature vectors are described in the following.

**Unary potential** $\alpha^\top \cdot \phi(x_j, h_j)$ : This potential function models the compatibility between $x_j$ and the part label $h_j$, i.e., how likely the patch $x_j$ is labeled as part $h_j$. It is parameterized as

$$\alpha^\top \cdot \phi(x_j, h_j) = \sum_{c \in \mathcal{H}} \alpha_c^\top \cdot \mathbf{1}_{\{h_j = c\}} \cdot [f^a(x_j) \ f^s(x_j)] \tag{3}$$

where we use $[f^a(x_j) \; f^s(x_j)]$ to denote the concatenation of two vectors $f^a(x_j)$ and $f^s(x_j)$. $f^a(x_j)$ is a feature vector describing the appearance of the patch $x_j$. In our case, $f^a(x_j)$ is simply the concatenation of four channels of the motion features at patch $x_j$, i.e., $f^a(x_j) = [Fb_x^+(x_j) \; Fb_x^-(x_j) \; Fb_y^+(x_j) \; Fb_y^-(x_j)]$. $f^s(x_j)$ is a feature vector describing the spatial location of the patch $x_j$. We discretize the whole image locations into $l$ bins, and $f^s(x_j)$ is a length $l$ vector of all zeros with a single one for the bin occupied by $x_j$. The parameter $\alpha_c$ can be interpreted as the measurement of compatibility between feature vector $[f^a(x_j) \; f^s(x_j)]$ and the part label $h_j = c$. The parameter $\alpha$ is simply the concatenation of $\alpha_c$ for all $c \in \mathcal{H}$.

**Unary potential** $\beta^\top \cdot \varphi(y, h_j)$ : This potential function models the compatibility between class label $y$ and part label $h_j$, i.e., how likely an image with class label $y$ contains a patch with part label $h_j$. It is parameterized as

$$\beta^\top \cdot \varphi(y, h_j) = \sum_{a \in \mathcal{Y}} \sum_{b \in \mathcal{H}} \beta_{a,b} \cdot \mathbf{1}_{\{y=a\}} \cdot \mathbf{1}_{\{h_j=b\}} \tag{4}$$

where $\beta_{a,b}$ indicates the compatibility between $y = a$ and $h_j = b$.

**Pairwise potential** $\gamma^\top \cdot \psi(y, h_j, h_k)$**:** This pairwise potential function models the compatibility between class label $y$ and a pair of part labels $(h_j, h_k)$, i.e., how likely an image with class label $y$ contains a pair of patches with part labels $h_j$ and $h_k$, where $(j, k) \in E$ corresponds to an edge in the graph. It is parameterized as

$$\gamma^\top \cdot \psi(y, h_j, h_k) = \sum_{a \in \mathcal{Y}} \sum_{b \in \mathcal{H}} \sum_{c \in \mathcal{H}} \gamma_{a,b,c} \cdot \mathbf{1}_{\{y=a\}} \cdot \mathbf{1}_{\{h_j=b\}} \cdot \mathbf{1}_{\{h_k=c\}} \tag{5}$$

where $\gamma_{a,b,c}$ indicates the compatibility of $y = a$, $h_j = b$ and $h_k = c$ for the edge $(j, k) \in E$.

**Root model** $\eta^\top \cdot \omega(y, \mathbf{x})$**:** The root model is a potential function that models the compatibility of class label $y$ and the large-scale global feature of the whole image. It is parameterized as

$$\eta^\top \cdot \omega(y, \mathbf{x}) = \sum_{a \in \mathcal{Y}} \eta_a^\top \cdot \mathbf{1}_{\{y=a\}} \cdot g(\mathbf{x}) \tag{6}$$

where $g(\mathbf{x})$ is a feature vector describing the appearance of the whole image. In our case, $g(\mathbf{x})$ is the concatenation of all the four channels of the motion features in the image, i.e., $g(\mathbf{x}) = [Fb_x^+ \; Fb_x^- \; Fb_y^+ \; Fb_y^-]$. $\eta_a$ can be interpreted as a root filter that measures the compatibility between the appearance of an image $g(\mathbf{x})$ and a class label $y = a$. And $\eta$ is simply the concatenation of $\eta_a$ for all $a \in \mathcal{Y}$.

The parameterization of $\Psi(y, \mathbf{h}, \mathbf{x})$ is similar to that used in object recognition [16]. But there are two important differences. First of all, our definition of the unary potential function $\phi(\cdot)$ encodes both appearance and spatial information of the patches. Secondly, we have a potential function $\omega(\cdot)$ describing the large scale appearance of the whole image. The representation in Quattoni et al. [16] only models local patches extracted from the image. This may be appropriate for object recognition. But for human action recognition, it is not clear that local patches can be sufficiently informative. We will demonstrate this experimentally in Sec. 4.

## 3  Learning and Inference

The model parameters $\theta$ are learned by maximizing the conditional log-likelihood on the training images:

$$\theta^* = \arg\max_\theta L(\theta) = \arg\max_\theta \sum_t \log p(y^t | \mathbf{x}^t; \theta) = \arg\max_\theta \sum_t \log \left( \sum_{\mathbf{h}} p(y^t, \mathbf{h} | \mathbf{x}^t; \theta) \right) \tag{7}$$

The objective function $L(\theta)$ in Quattoni et al.[16] also has a regularization term $\frac{-1}{2\sigma^2} ||\theta||^2$. In our experiments, we find that the regularization does not seem to have much effect on the final results, so we will use the un-regularized version. Different from conditional random field (CRF) [11], the objective function $L(\theta)$ of hCRF is not concave, due to the hidden variables $\mathbf{h}$. But we can still use

gradient ascent to find $\theta$ that is locally optimal. The gradient of the log-likelihood with respect to the $t$-th training image $(\mathbf{x}^t, y^t)$ can be calculated as:

$$\frac{\partial L^t(\theta)}{\partial \alpha} = \sum_{j \in V} \left[ \mathbb{E}_{p(h_j|y^t,\mathbf{x}^t;\theta)} \phi(x_j^t, h_j) - \mathbb{E}_{p(h_j,y|\mathbf{x}^t;\theta)} \phi(x_j^t, h_j) \right]$$

$$\frac{\partial L^t(\theta)}{\partial \beta} = \sum_{j \in V} \left[ \mathbb{E}_{p(h_j|y^t,\mathbf{x}^t;\theta)} \varphi(h_j, y^t) - \mathbb{E}_{p(h_j,y|\mathbf{x}^t;\theta)} \varphi(h_j, y) \right]$$

$$\frac{\partial L^t(\theta)}{\partial \gamma} = \sum_{(j,k) \in E} \left[ \mathbb{E}_{p(h_j,h_k|y^t,\mathbf{x}^t;\theta)} \psi(y^t, h_j, h_k) - \mathbb{E}_{p(h_j,h_k,y|\mathbf{x}^t;\theta)} \psi(y, h_j, h_k) \right]$$

$$\frac{\partial L^t(\theta)}{\partial \eta} = \omega(y^t, \mathbf{x}^t) - \mathbb{E}_{p(y|\mathbf{x}^t;\theta)} \omega(y, \mathbf{x}^t) \tag{8}$$

Assuming the edges $E$ form a tree, the expectations in Eq. 8 can be calculated in $O(|\mathcal{Y}||E||\mathcal{H}|^2)$ time using belief propagation.

Now we describe several details about how the above ideas are implemented.

**Learning root filter** $\eta$: Given a set of training images $(\mathbf{x}^t, y^t)$, we firstly learn the root filter $\eta$ by solving the following optimization problem:

$$\eta^* = \arg \max_{\eta} \sum_{t} \log \mathcal{L}(y^t|\mathbf{x}^t; \eta) = \arg \max_{\eta} \sum_{t} \log \frac{\exp \left( \eta^\top \cdot \omega(y^t, \mathbf{x}^t) \right)}{\sum_y \exp \left( \eta^\top \cdot \omega(y, \mathbf{x}^t) \right)} \tag{9}$$

In other words, $\eta^*$ is learned by only considering the feature vector $\omega(\cdot)$. We then use $\eta^*$ as the starting point for $\eta$ in the gradient ascent (Eq. 8). Other parameters $\alpha$, $\beta$, $\gamma$ are initialized randomly.

**Patch initialization**: We use a simple heuristic similar to that used in [6] to initialize ten salient patches on every training image from the root filter $\eta^*$ trained above. For each training image $I$ with class label $a$, we apply the root filter $\eta_a$ on $I$, then select an rectangle region of size $5 \times 5$ in the image that has the most positive energy. We zero out the weights in this region and repeat until ten patches are selected. Figure 4(a) shows examples of the patches found in some images. The tree $G = (V, E)$ is formed by running a minimum spanning tree algorithm over the ten patches.

**Inference**: During testing, we do not know the class label of a given test image, so we cannot use the patch initialization described above to initialize the patches, since we do not know which root filter to use. Instead, we run root filters from all the classes on a test image, then calculate the probabilities of all possible instantiations of patches under our learned model, and classify the image by picking the class label that gives the maximum of the these probabilities. In other words, for a testing image with motion descriptor $\mathbf{x}$, we first obtain $|\mathcal{Y}|$ instances $\{\mathbf{x}^{(1)}, \mathbf{x}^{(2)}, ..., \mathbf{x}^{(|\mathcal{Y}|)}\}$, where each $\mathbf{x}^{(k)}$ is obtained by initializing the patches on $\mathbf{x}$ using the root filter $\eta_k$. The final class label $y^*$ of $\mathbf{x}$ is obtained as $y^* = \arg \max_y \left[ \max\{p(y|\mathbf{x}^{(1)}; \theta), p(y|\mathbf{x}^{(2)}; \theta), ..., p(y|\mathbf{x}^{(|\mathcal{Y}|)}; \theta)\} \right]$.

## 4  Experiments

We test our algorithm on two publicly available datasets that have been widely used in action recognition: Weizmann human action dataset [2], and KTH human motion dataset [17]. Performance on these benchmarks is saturating – state-of-the-art approaches achieve near-perfect results. We show our method achieves results comparable to the state-of-the-art, and more importantly that our extended hCRF model significantly outperforms a direct application of the original hCRF model [16].

**Weizmann dataset:** The Weizmann human action dataset contains 83 video sequences showing nine different people, each performing nine different actions: running, walking, jumping-jack, jumping-forward-on-two-legs,jumping-in-place-on-two-legs, galloping-sideways, waving-two-hands, waving-one-hand, bending. We track and stabilize the figures using the background subtraction masks that come with this dataset.

We randomly choose videos of five subjects as training set, and the videos in the remaining four subjects as test set. We learn three hCRF models with different sizes of possible part labels, $|\mathcal{H}| = 6, 10, 20$. Our model classifies every frame in a video sequence (i.e., per-frame classification), but

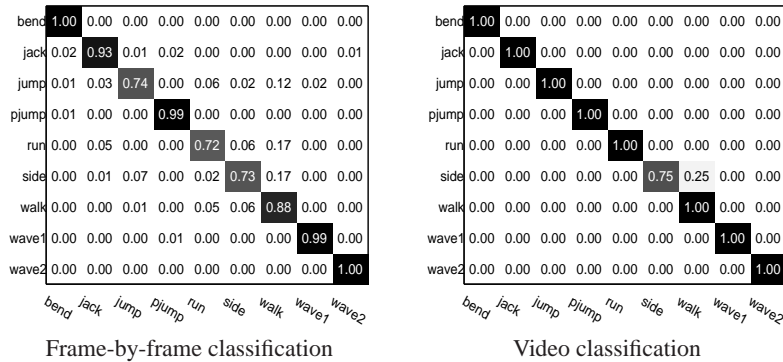

Frame-by-frame classification                     Video classification

Figure 3: Confusion matrices of classification results on Weizmann dataset. Horizontal rows are ground truths, and vertical columns are predictions.

| method | root model | local hCRF | | | our approach | | |
|---|---|---|---|---|---|---|---|
| | | $|\mathcal{H}| = 6$ | $|\mathcal{H}| = 10$ | $|\mathcal{H}| = 20$ | $|\mathcal{H}| = 6$ | $|\mathcal{H}| = 10$ | $|\mathcal{H}| = 20$ |
| per-frame | 0.7470 | 0.5722 | 0.6656 | 0.6383 | **0.8682** | **0.9029** | **0.8557** |
| per-video | 0.8889 | 0.5556 | 0.6944 | 0.6111 | **0.9167** | **0.9722** | **0.9444** |

Table 1: Comparison of two baseline systems with our approach on Weizmann dataset.

we can also obtain the class label for the whole video sequence by the majority voting of the labels of its frames (i.e., per-video classification). We show the confusion matrix with $|\mathcal{H}| = 10$ for both per-frame and per-video classification in Fig. 3.

We compare our system to two baseline methods. The first baseline (root model) only uses the root filter $\eta^\top \cdot \omega(y, \mathbf{x})$, which is simply a discriminative version of Efros et al. [5]. The second baseline (local hCRF) is a direct application of the original hCRF model [16]. It is similar to our model, but without the root filter $\eta^\top \cdot \omega(y, \mathbf{x})$, i.e., local hCRF only uses the root filter to initialize the salient patches, but does not use it in the final model. The comparative results are shown in Table 1. Our approach significantly outperforms the two baseline methods. We also compare our results(with $|\mathcal{H}| = 10$) with previous work in Table 2. Note [2] classifies space-time cubes. It is not clear how it can be compared with other methods that classify frames or videos. Our result is significantly better than [13], and comparable to [8]. Although we accept the fact that the comparison is not completely fair, since [13] does not use any tracking or background subtraction.

We visualize the learned parts in Fig. 4(a). Each patch is represented by a color that corresponds to the most likely part label of that patch. We also visualize the root filters applied on these images in Fig. 4(b).

**KTH dataset:** The KTH human motion dataset contains six types of human actions (walking, jogging, running, boxing, hand waving and hand clapping) performed several times by 25 subjects in four different scenarios: outdoors, outdoors with scale variation, outdoors with different clothes and indoors. We first run an automatic preprocessing step to track and stabilize the video sequences, so that all the figures appear in the center of the field of view.

We split the videos roughly equally into training/test sets and randomly sample 10 frames from each video. The confusion matrices (with $|\mathcal{H}| = 10$) for both per-frame and per-video classification are

| | per-frame(%) | per-video(%) | per-cube(%) |
|---|---|---|---|
| Our method | 90.3 | 97.2 | N/A |
| Jhuang et al. [8] | N/A | 98.8 | N/A |
| Niebles & Fei-Fei [13] | 55 | 72.8 | N/A |
| Blank et al. [2] | N/A | N/A | 99.64 |

Table 2: Comparison of classification accuracy with previous work on the Weizmann dataset.

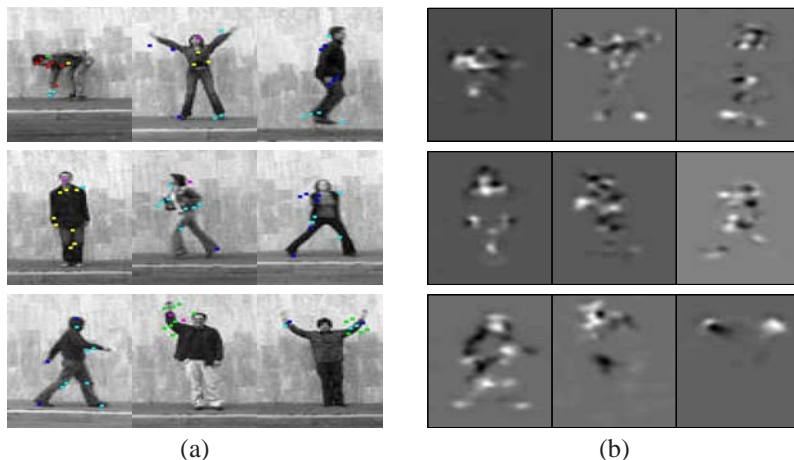

|(a)|(b)|

Figure 4: (a) Visualization of the learned parts. Patches are colored according to their most likely part labels. Each color corresponds to a part label. Some interesting observations can be made. For example, the part label represented by red seems to correspond to the "moving down" patterns mostly observed in the "bending" action. The part label represented by green seems to correspond to the motion patterns distinctive of "hand-waving" actions; (b) Visualization of root filters applied on these images. For each image with class label $c$, we apply the root filter $\eta_c$. The results show the filter responses aggregated over four motion descriptor channels. Bright areas correspond to positive energies, i.e., areas that are discriminative for this class.

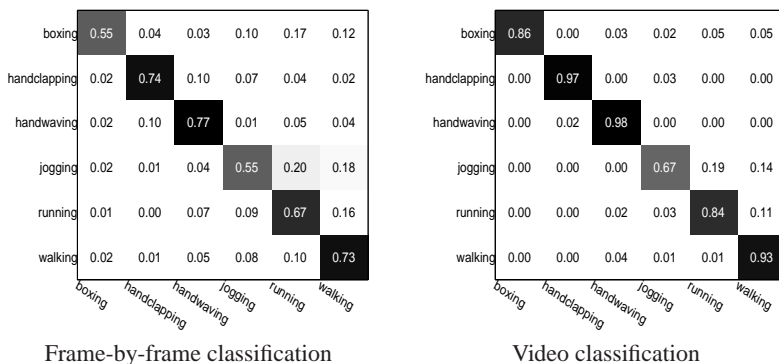

Frame-by-frame classification          Video classification

Figure 5: Confusion matrices of classification results on KTH dataset. Horizontal rows are ground truths, and vertical columns are predictions.

shown in Fig. 5. The comparison with the two baseline algorithms is summarized in Table 3. Again, our approach outperforms the two baselines systems.

The comparison with other approaches is summarized in Table 4. We should emphasize that we do not attempt a direct comparison, since different methods listed in Table 4 have all sorts of variations in their experiments (e.g., different split of training/test data, whether temporal smoothing is used, whether per-frame classification can be performed, whether tracking/background subtraction is used, whether the whole dataset is used etc.), which make it impossible to directly compare them. We provide the results only to show that our approach is comparable to the state-of-the-art.

| method | root model | local hCRF | | | our approach | | |
|--------|-----------|-----------|-----------|-----------|-----------|-----------|-----------|
| | | $|\mathcal{H}| = 6$ | $|\mathcal{H}| = 10$ | $|\mathcal{H}| = 20$ | $|\mathcal{H}| = 6$ | $|\mathcal{H}| = 10$ | $|\mathcal{H}| = 20$ |
| per-frame | 0.5377 | 0.4749 | 0.4452 | 0.4282 | **0.6633** | **0.6698** | **0.6444** |
| per-video | 0.7339 | 0.5607 | 0.5814 | 0.5504 | **0.7855** | **0.8760** | **0.7512** |

Table 3: Comparison of two baseline systems with our approach on KTH dataset.

| methods | accuracy(%) |
|---|---|
| Our method | **87.60** |
| Jhuang et al. [8] | **91.70** |
| Nowozin et al. [15] | 87.04 |
| Niebles et al. [14] | 81.50 |
| Dollár et al. [4] | 81.17 |
| Schuldt et al. [17] | 71.72 |
| Ke et al. [9] | 62.96 |

Table 4: Comparison of per-video classification accuracy with previous approaches on KTH dataset.

## 5 Conclusion

We have presented a discriminatively learned part model for human action recognition. Unlike previous work [10], our model does not require manual specification of the parts. Instead, the parts are initialized by a learned root filter. Our model combines both large-scale features used in global templates and local patch features used in bag-of-words models. Our experimental results show that our model is quite effective in recognizing actions. The results are comparable to the state-of-the-art approaches. In particular, we show that the combination of large-scale features and local patch features performs significantly better than using either of them alone.

## References

[1] A. C. Berg, T. L. Berg, and J. Malik. Shape matching and object recognition using low distortion correspondence. In *IEEE CVPR*, 2005.

[2] M. Blank, L. Gorelick, E. Shechtman, M. Irani, and R. Basri. Actions as space-time shapes. In *IEEE ICCV*, 2005.

[3] N. Dalal and B. Triggs. Histogram of oriented gradients for human detection. In *IEEE CVPR*, 2005.

[4] P. Dollár, V. Rabaud, G. Cottrell, and S. Belongie. Behavior recognition via sparse spatio-temporal features. In *VS-PETS Workshop*, 2005.

[5] A. A. Efros, A. C. Berg, G. Mori, and J. Malik. Recognizing action at a distance. In *IEEE ICCV*, 2003.

[6] P. Felzenszwalb, D. McAllester, and D. Ramanan. A discriminatively trained, multiscale, deformable part model. In *IEEE CVPR*, 2008.

[7] P. F. Felzenszwalb and D. P. Huttenlocher. Pictorial structures for object recognition. *IJCV*, 61(1):55–79, January 2003.

[8] H. Jhuang, T. Serre, L. Wolf, and T. Poggio. A biologically inspired system for action recognition. In *IEEE ICCV*, 2007.

[9] Y. Ke, R. Sukthankar, and M. Hebert. Efficient visual event detection using volumetric features. In *IEEE ICCV*, 2005.

[10] Y. Ke, R. Sukthankar, and M. Hebert. Event detection in crowded videos. In *IEEE ICCV*, 2007.

[11] J. Lafferty, A. McCallum, and F. Pereira. Conditional random fields: Probabilistic models for segmenting and labeling sequence data. In *ICML*, 2001.

[12] B. D. Lucas and T. Kanade. An iterative image registration technique with an application to stereo vision. In *Proc. DARPA Image Understanding Workshop*, 1981.

[13] J. C. Niebles and L. Fei-Fei. A hierarchical model of shape and appearance for human action classification. In *IEEE CVPR*, 2007.

[14] J. C. Niebles, H. Wang, and L. Fei-Fei. Unsupervised learning of human action categories using spatial-temporal words. In *BMVC*, 2006.

[15] S. Nowozin, G. Bakir, and K. Tsuda. Discriminative subsequence mining for action classification. In *IEEE ICCV*, 2007.

[16] A. Quattoni, M. Collins, and T. Darrell. Conditional random fields for object recognition. In *NIPS 17*, 2005.

[17] C. Schuldt, L. Laptev, and B. Caputo. Recognizing human actions: a local SVM approach. In *IEEE ICPR*, 2004.

[18] J. Sivic, B. C. Russell, A. A. Efros, A. Zisserman, and W. T. Freeman. Discovering objects and their location in images. In *IEEE ICCV*, 2005.

[19] S. B. Wang, A. Quattoni, L.-P. Morency, D. Demirdjian, and T. Darrell. Hidden conditional random fields for gesture recognition. In *IEEE CVPR*, 2006.
